# Vertex Identification in High Energy Physics Experiments

**Gideon Dror**[*]
Department of Computer Science
The Academic College of Tel-Aviv-Yaffo, Tel Aviv 64044, Israel

**Halina Abramowicz**[†]  **David Horn**[‡]
School of Physics and Astronomy
Raymond and Beverly Sackler Faculty of Exact Sciences
Tel-Aviv University, Tel Aviv 69978, Israel

## Abstract

In High Energy Physics experiments one has to sort through a high flux of events, at a rate of tens of MHz, and select the few that are of interest. One of the key factors in making this decision is the location of the vertex where the interaction, that led to the event, took place. Here we present a novel solution to the problem of finding the location of the vertex, based on two feedforward neural networks with fixed architectures, whose parameters are chosen so as to obtain a high accuracy. The system is tested on simulated data sets, and is shown to perform better than conventional algorithms.

## 1  Introduction

An event in High Energy Physics (HEP) is the experimental result of an interaction during the collision of particles in an accelerator. The result of this interaction is the production of tens of particles, each of which is ejected in a different direction and energy. Due to the quantum mechanical effects involved, the events differ from one another in the number of particles produced, the types of particles, and their energies. The trajectories of produced particles are detected by a very large and sophisticated detector.

---
[*]gideon@server.mta.ac.il
[†]halina@post.tau.ac.il
[‡]horn@neuron.tau.ac.il

Events are typically produced at a rate of 10 MHz, in conjunction with a data volume of up to 500 kBytes per event. The signal is very small, and is selected from the background by multilevel triggers that perform filtering either through hardware or software. In the present paper we confront one problem that is of interest in these experiments and is part of the triggering consideration. This is the location of the vertex of the interaction. To be specific we will use a simulation of data collected by the central tracking detector [1] of the ZEUS experiment [2] at the HEP laboratory DESY in Hamburg, Germany. This detector, placed in a magnetic field, surrounds the interaction point and is sensitive to the path of charged particles. It has a cylindrical shape around the axis, $z$, where the interaction between the incoming particles takes place. The challenge is to find an efficient and fast method to extract the exact location of the vertex along this axis.

## 2   The Input Data

An example of an event, projected onto the $z = 0$ plane, is shown in Figure 1. Only the information relevant to triggering is used and displayed. The relevant points, which denote hits by the outgoing particles on wires in the detector, form five rings due to the concentric structure of the detector. Several slightly curved particle tracks emanating from the origin, which is marked with a + sign, and crossing all five rings, can easily be seen. Each track is made of 30-40 data points. All tracks appear in this projection as arcs, and indeed, when viewed in 3 dimensions, every particle follows a helical trajectory due to the solenoidal magnetic field in the detector.

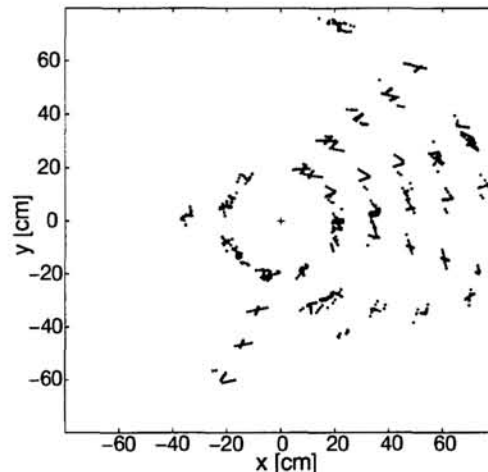

Figure 1: A typical event projected onto the $z = 0$ plane. The dots, or hits, have a two-fold ambiguity in the determination of the $xy$ coordinates through which the particle has moved. The correct solutions lie on curved tracks that emanate from the origin.

Each physical hit is represented twice in Fig. 1 due to an inherent two-fold ambiguity in the determination of its $xy$ coordinates. The correct solutions form curved tracks emanating from the origin. Some of those can be readily seen in the data. Due to the limited time available for decision making at the trigger level, the $z$ coordinate is obtained from the difference in arrival times of a pulse at both ends of the CTD and is available for only a fraction of these points. The hit resolution in $xy$ is $\sim 230\,\mu$m, while that of $z$-by-timing is $\simeq 4$ cm. The quality of the $z$ coordinate

information is exemplified in figure 2. Figure 2(a) shows points forming a track of
a single particle on the $z = 0$ projection. Since the corresponding track forms a
helix with small curvature, one expects a linear dependence of the $z$ coordinate of
the hits on their radial position, $r = \sqrt{x^2 + y^2}$. Figure 2(b) compares the values of
$r$ with the measured $z$ values for these points. The scatter of the data around the
linear regression fit is considerable.

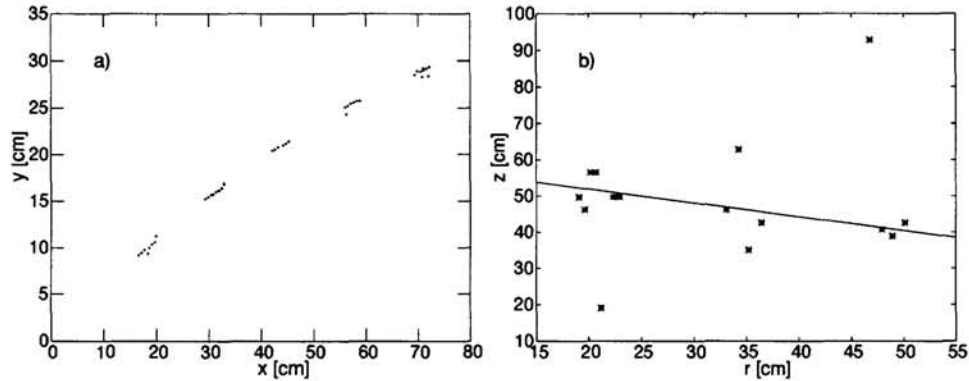

Figure 2: A typical example of uncertainties in the measured $z$ values: (a) a single
track taken from the event shown in figure 1, (b) the $z$ coordinate vs $r = \sqrt{x^2 + y^2}$
the distance from the $z$ axis for the data points shown in (a). The full line is a
linear regression fit.

## 3   The Network

Our network is based on step-wise changes in the representation of the data, moving
from the input points, to local line segments and to global arcs. The nature of
the data and the problem suggest it is best to separate the treatment of the $xy$
coordinates from that of the $z$ coordinate. Two parallel networks which perform
entirely different computations, form our final system. The first network, which
handles the $xy$ information is responsible for constructing arcs that correctly identify
some of the particle tracks in the event. The second network uses this information
to evaluate the $z$ location of the point where all tracks meet.

### 3.1   Arc Identification Network

The arc identification network processes information in a fashion akin to the method
visual information is processed by the primary visual system [3].

The input layer for this network is made of a large number of neurons (several tens
of thousands) and corresponds to the function of the retina. Each input neuron
has its distinct receptive field. The sum of all fields covers completely the relevant
domain in the $xy$ plane. This domain has 5 concentric rings, which show up in
figure 1. The total area of the rings is about $5000$ cm$^2$, and covering it with $100000$
input neurons leads to satisfactory resolution. A neuron in the input level fires
when a hit is present in its receptive field. We shall label each input neuron by the
$(xy)$ coordinates of the center of its receptive field.

Neurons of the second layer are line segment detectors. Each second layer neuron
is labeled by $(XY\alpha)$, where $(X, Y)$ are the coordinates of the center of the segment

and $\alpha$ denotes its orientation. The activation of second layer neurons is given by

$$V_{XY\alpha} = g(\sum_{xy} J_{XY\alpha,xy} V_{xy} - \theta_2) \,, \tag{1}$$

where

$$J_{XY\alpha,xy} = \begin{cases} 1 & \text{if } r_\perp < 0.5\,\text{cm} \wedge r_\| < 2\,\text{cm} \\ -1 & \text{if } 0.5\,\text{cm} < r_\perp < 1\,\text{cm} \wedge r_\| < 2\,\text{cm} \\ 0 & \text{otherwise} \end{cases} \tag{2}$$

and $g(x)$ is the standard Heaviside step function. $r_\|$ and $r_\perp$ are the parallel and perpendicular distances between $(X, Y)$ and $(x, y)$ with respect to the axis of the line segment, defined by $\alpha$. It is important to note that at this level, values of the threshold $\theta_2$ which are slightly lower than optimum are preferable, taking the risk of obtaining superfluous line segments in order to reduce the probability of missing one. Superfluous line segments are filtered out very efficiently in higher layers.

Figure 3 represents the output of the second layer neurons for the input illustrated by the event of figure 1. An active second layer neuron $(XY\alpha)$ is represented in this figure by a line segment centered at the point $(X, Y)$ making an angle $\alpha$ with the $x$ axis. The length of the line segments is immaterial and was chosen only for the purpose of visual clarity.

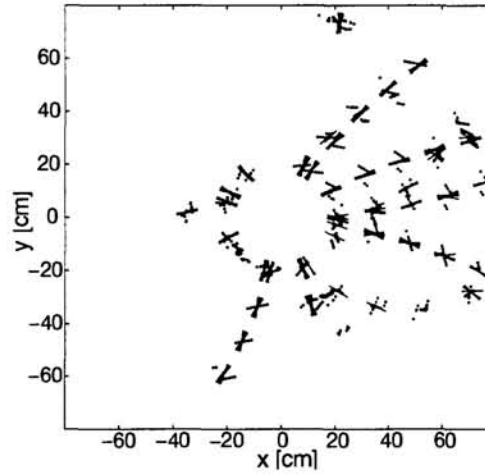

Figure 3: Representation of the activity of second layer neurons $XY\alpha$ for the input of figure 1 taken by plotting the appropriate line segments in the $xy$ plane. At some $XY$ locations several line segments with different directions occur due to the rather low threshold parameter used, $\theta_2 = 4$.

Neurons of the third layer transform the representation of local line segments into local arc segments. An arc which passes through the origin is uniquely defined by its radius of curvature $R$ and its slope at the origin. Thus, each third layer neuron is labeled by $\kappa\theta i$, where $|\kappa| = 1/R$ is the curvature and the sign of $\kappa$ determines the orientation of the arc. $1 \le i \le 5$ is an index which relates each arc segment to the ring it belongs to.

The mapping between second and third layers is based on a winner-take-all mechanism. Namely, for a given local arc segment, we take the arc segment which is closest to being tangent to the local arc segment.

Denoting the average radius of the ring $i$ ( i=1,2,...5) by $r_i$ and using $\beta_i = \sin^{-1}(\frac{\kappa r_i}{2})$

the final expression for the activation of the third layer neurons is

$$V_{\kappa\theta i} = \max_{\delta < 3} e^{-\delta^2} \cos^2(\theta - 2\beta_i - \alpha) \,, \qquad (3)$$

where $\delta = \delta(X, Y, \kappa, \theta, i) = \sqrt{(X - r_i \cos(\theta - \beta_i))^2 + (Y - r_i \sin(\theta - \beta_i))^2}$ is simply the distance of the center of the receptive field of the $(XY\alpha)$ neuron to the $(\kappa\theta)$ arc.

The fourth layer is the last one in the arc identification network. Neurons belonging to this layer are global arc detectors. In other words, they detect projected tracks on the $z = 0$ plane. A fourth level neuron is denoted by $\kappa\theta$, where $\kappa$ and $\theta$ have the previous meaning, now describing global arcs. Fourth layer neurons are connected to third layer neurons in a simple fashion,

$$V_{\kappa\theta} = g\Big(\sum_{\kappa'\theta'i} \delta_{\kappa,\kappa'} \delta_{\theta,\theta'} V_{\kappa'\theta'i} - \theta_4\Big) \,. \qquad (4)$$

Figure 4 represents the activity of fourth layer neurons. Each active neuron $\kappa\theta$ is equivalent in the $xy$ plane to one arc appearing in the figure.

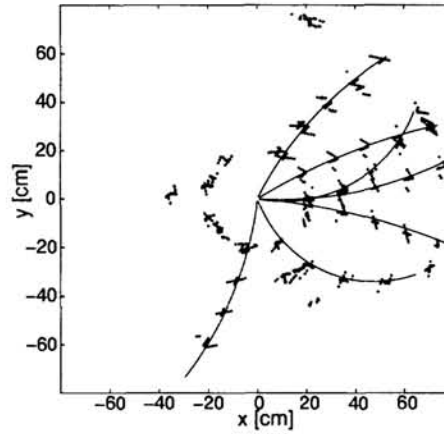

Figure 4: Representation of the activity of fourth layer neurons $\kappa\theta$ for the input of figure 1 taken by plotting the appropriate arcs in the $xy$ plane. The arcs are not precisely congruent to the activity of the input layer which is also shown, due to the finite widths which were used, $\Delta\kappa = 0.004$ and $\Delta\theta = \pi/20$. This figure was produced with $\theta_4 = 3$.

## 3.2   z Location Network

The architecture of the second network has a structure which is identical to the first one, although its computational task is different. We will use an identical labeling system for its neurons, but denote their activities by $v_{xy}$. The latter will assume continuous values in this network.

A first layer neuron of the z-location network receives its input from the same receptive field as its corresponding neuron in the first network. Its value, $v_{xy}$, is the mean value of the $z$ values of the points within its receptive field. If no $z$ values are available for these points, a null value is assigned to it.

The second layer neurons compute the mean value $v_{XY\alpha} = \langle v_{xy} \rangle$ of the $z$ coordinate of the first layer neurons in their receptive field, averaging over all neurons within

the section
$$\left\{ xy \,\middle|\, |(x - X)\sin\alpha - (y - Y)\cos\alpha| < 0.5\,\mathrm{cm} \wedge (x - X)^2 + (y - Y)^2 < 4\,\mathrm{cm}^2 \right\} \ ,$$
which corresponds to the excitatory part of the synaptic connections of equation (2). If null values appear within that section they are disregarded by the averaging procedure. If all values are null, $v_{XY\alpha}$ is assigned a null value too. This $z$ averaging procedure is similarly propagated to the third layer neurons.

The fourth layer neurons evaluate the $z$ value of the origin of each arc identified by the first network. This is performed by a simple linear extrapolation. The final $z$ estimate of the vertex, $z_{net}$, which should be the common origin of all arcs, is calculated by averaging the outputs of all active fourth layer neurons.

## 4 Results

In order to test the network, we ran it over a set of 1000 events generated by a Monte-Carlo simulator as well as over a sample of physical events taken from the ZEUS experiment at the HEP laboratory DESY in Hamburg. For the former set we compared the estimate of the net $z_{net}$ with the nominal location of the vertex $z$, whereas for the real events in the latter set, we compared it with an estimate $z_{rec}$ obtained by full reconstruction algorithm, which runs off-line and uses all available data. Results of the two tests can be compared since it is well established that the result of the full reconstruction algorithm is within 1 mm from the exact location of the vertex.

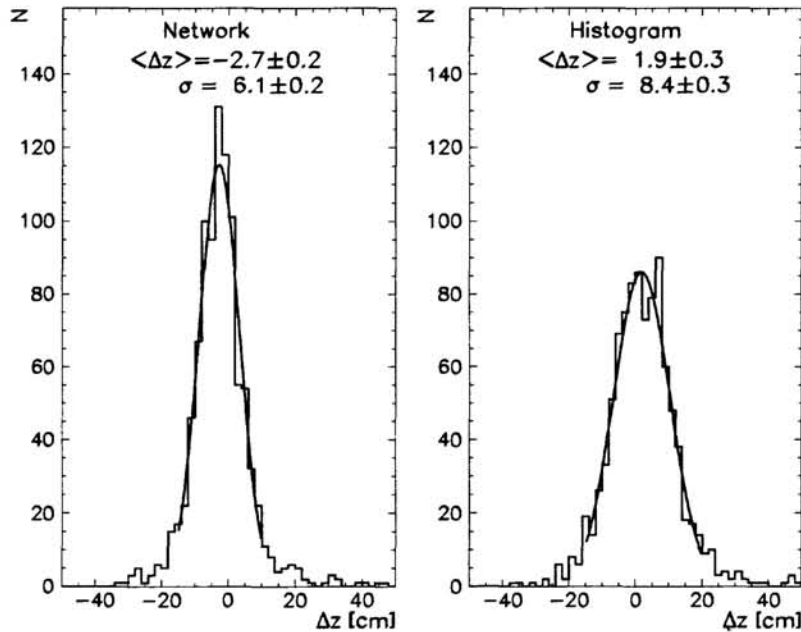

Figure 5: Distribution of $\Delta z = z_{estimate} - z_{exact}$ values for two types of estimates, (a) the one proposed in this paper and (b) the one based on a commonly used histogram method.

We also compared our results with those of an algorithmic method used for triggering at ZEUS [4]. We shall refer to this method as the 'histogram method'. The performance of the two methods was compared on a sample of 1000 Monte-Carlo events. The network was unable to get an estimate for 16 events from the set, as compared with 15 for the histogram method (15 of those events were common

failures). In Figure 5 we compare the distributions of $\Delta z = z_{net} - z_{exact}$ and $\Delta z = z_{hist} - z_{exact}$ for the sample of Monte-Carlo events, where $z_{exact}$ is the generated location of the vertex. Both methods lead to small biases, $-2.7$ cm for $z_{net}$ and 1.9 cm for $z_{hist}$. The resolution, as obtained from a Gaussian fit, was found to be better for the network approach ($\sigma = 6.1$ cm) as compared to the histogram method ($\sigma = 8.4$ cm). In addition, it should be noted that the histogram method yields discrete results, with a step of 10 cm, whereas the current method gives continuous values. This can be of great advantage for further processing. Note that off-line, after using the whole CTD information, the resolution is better than 1 mm.

## 5   Discussion

We have described a feedforward double neural network that performs a task of pattern identification by thresholding and selecting subsets of data on which a simple computation can lead to the final answer. The network uses a fixed architecture, which allows for its implementation in hardware, crucial for fast triggering purposes.

The basic idea of using a fixed architecture that is inspired by the way our brain processes visual information, is similar to the the raison d'être of the orientation selective neural network employed by [5]. The latter was based on orientation selective cells only, which were sufficient to select linear tracks that are of interest in HEP experiments. Here we develop an arc identification method, following similar steps. Both methods can also be viewed as generalizations of the Hough transform [6] that was originally proposed for straight line identification and may be regarded as a basic element of pattern recognition problems [7]. Neither [5] nor our present proposal were considered by previous neural network analyses of HEP data [8]. The results that we have obtained are very promising. We hope that they open the possibility for a new type of neural network implementation in triggering devices of HEP experiments.

### Acknowledgments

We are indebted to the ZEUS Collaboration whose data were used for this study. This research was partially supported by the Israel National Science Foundation.

## References

[1] B. Foster et al., Nuclear Instrum. and Methods in Phys. Res. A338 (1994) 254.

[2] ZEUS Collab., The ZEUS Detector, Status Report 1993, DESY 1993; M. Derrick et al., Phys. Lett. B 293 (1992) 465.

[3] D. H. Hubel and T. N. Wiesel, J. Physiol. 195 (1968) 215.

[4] A. Quadt, MSc thesis, University of Oxford (1997).

[5] H. Abramowicz, D. Horn, U. Naftaly and C. Sahar-Pikielny, Nuclear Instrum. and Methods in Phys. Res. A378 (1996) 305; *Advances in Neural Information Processing Systems 9*, eds. M. C. Mozer, M. J. Jordan and T. Petsche, MIT Press 1997, pp. 925–931.

[6] P. V. Hough, "Methods and means to recognize complex patterns", U.S. patent 3.069.654.

[7] R. O. Duda and P. E. Hart, "Pattern classification and scene analysis", Wiley, New York, 1973.

[8] B. Denby, Neural Computation, 5 (1993) 505.